# Sparse Estimation with Structured Dictionaries

**David P. Wipf** *
Visual Computing Group
Microsoft Research Asia
davidwipf@gmail.com

## Abstract

In the vast majority of recent work on sparse estimation algorithms, performance has been evaluated using ideal or quasi-ideal dictionaries (e.g., random Gaussian or Fourier) characterized by unit $\ell_2$ norm, incoherent columns or features. But in reality, these types of dictionaries represent only a subset of the dictionaries that are actually used in practice (largely restricted to idealized compressive sensing applications). In contrast, herein sparse estimation is considered in the context of structured dictionaries possibly exhibiting high coherence between arbitrary groups of columns and/or rows. Sparse penalized regression models are analyzed with the purpose of finding, to the extent possible, regimes of dictionary invariant performance. In particular, a Type II Bayesian estimator with a dictionary-dependent sparsity penalty is shown to have a number of desirable invariance properties leading to provable advantages over more conventional penalties such as the $\ell_1$ norm, especially in areas where existing theoretical recovery guarantees no longer hold. This can translate into improved performance in applications such as model selection with correlated features, source localization, and compressive sensing with constrained measurement directions.

## 1 Introduction

We begin with the generative model

$$Y = \Phi X_0 + \mathcal{E}, \tag{1}$$

where $\Phi \in \mathbb{R}^{n \times m}$ is a dictionary of basis vectors or features, $X_0 \in \mathbb{R}^{m \times t}$ is a matrix of unknown coefficients we would like to estimate, $Y \in \mathbb{R}^{n \times t}$ is an observed signal matrix, and $\mathcal{E}$ is a noise matrix with iid elements distributed as $\mathcal{N}(0, \lambda)$. The objective is to estimate the unknown generative $X_0$ under the assumption that it is row-sparse, meaning that many *rows* of $X_0$ have zero norm. The problem is compounded considerably by the additional assumption that $m > n$, meaning the dictionary $\Phi$ is overcomplete. When $t = 1$, this then reduces to the canonical sparse estimation of a coefficient vector with mostly zero-valued entries or minimal $\ell_0$ norm [7]. In contrast, estimation of $X_0$ with $t > 1$ represents the more general simultaneous sparse approximation problem [6, 15] relevant to numerous applications such as compressive sensing and multi-task learning [9, 16], manifold learning [13], array processing [10], and functional brain imaging [1]. We will consider both scenarios herein but will primarily adopt the more general notation of the $t > 1$ case.

One possibility for estimating $X_0$ involves solving

$$\min_{X} \|Y - \Phi X\|_{\mathcal{F}}^2 + \lambda d(X), \quad \lambda > 0, \quad d(X) \triangleq \sum_{i=1}^{m} \mathcal{I}\left[\|\boldsymbol{x}_{i\cdot}\| > 0\right], \tag{2}$$

where the indicator function $\mathcal{I}\left[\|\boldsymbol{x}\| > 0\right]$ equals one if $\|\boldsymbol{x}\| > 0$ and equals zero otherwise ($\|\boldsymbol{x}\|$ is an arbitrary vector norm). $d(X)$ penalizes the number of rows in $X$ that are not equal to zero;

for nonzero rows there is no additional penalty for large magnitudes. Moreover, it reduces to the $\ell_0$ norm when $t = 1$, i.e., $d(\boldsymbol{x}) = \|\boldsymbol{x}\|_0$, or a count of the nonzero elements in the vector $\boldsymbol{x}$. Note that to facilitate later analysis, we define $\boldsymbol{x}_{\cdot i}$ as the $i$-th column of matrix $X$ while $\boldsymbol{x}_{i\cdot}$ represents the $i$-th row. For theoretical inquiries or low-noise environments, it is often convenient to consider the limit as $\lambda \to 0$, in which case (2) reduces to

$$\min_X d(X), \qquad \text{s.t. } \Phi X_0 = \Phi X. \tag{3}$$

Unfortunately, solving either (2) or (3) involves a combinatorial search and is therefore not tractable in practice. Instead, a family of more convenient sparse penalized regression cost functions are reviewed in Section 2. In particular, we discuss conventional *Type I* sparsity penalties, such as the $\ell_1$ norm and the $\ell_{1,2}$ mixed norm, and a *Type II* empirical Bayesian alternative characterized by dictionary dependency. When the dictionary $\Phi$ is incoherent, meaning the columns are roughly orthogonal to one another, then certain Type I selections are well-known to produce good approximations of $X_0$ via efficient implementations. However, as discussed in Section 3, more structured dictionary types can pose difficulties. In Section 4 we analyze the underlying cost functions of Type I and Type II, and demonstrate that the later maintains several properties that suggest it will be robust to highly structured dictionaries. Brief empirical comparisons are presented in Section 5.

## 2   Estimation via Sparse Penalized Regression

Directly solving either (2) or (3) is intractable, so a variety of approximate methods have been proposed. Many of these can be viewed simply as regression with a sparsity penalty convenient for optimization purposes. The general regression problem we consider here involves solving

$$\min_X \|Y - \Phi X\|_{\mathcal{F}}^2 + \lambda g(X), \tag{4}$$

where $g$ is some penalty function of the row norms. Type I methods use a separable penalty of the form

$$g^{(I)}(X) = \sum_i h\left(\|\boldsymbol{x}_{i\cdot}\|_2\right), \tag{5}$$

where $h$ is a non-decreasing, typically concave function.[1]  Common examples include $h(z) = z^p, p \in (0, 1]$ [11] and $h(z) = \log(z + \alpha), \alpha \geq 0$ [4]. The parameters $p$ and $\alpha$ are often heuristically selected on an application-specific basis. In contrast, Type II methods, with origins as empirical Bayesian estimators, implicitly utilize a more complicated penalty function that can only be expressed in a variational form [18]. Herein we will consider the selection

$$g^{(II)}(X) \triangleq \min_{\Gamma \succeq 0} \text{Tr}\left[X^T \Gamma^{-1} X\right] + t \log\left|\alpha I + \Phi \Gamma \Phi^T\right|, \quad \alpha \geq 0, \tag{6}$$

where $\Gamma$ is a diagonal matrix of non-negative variational parameters [14, 18]. While less transparent than Type I, it has been shown that (6) is a concave non-decreasing function of each row norm of $X$, hence it promotes row sparsity as well. Moreover, the dictionary-dependency of this penalty appears to be the source of some desirable invariance properties as discussed in Section 4. Analogous to (3), for analytical purposes all of these methods can be reduced as $\lambda \to 0$ to solving

$$\min_X g(X) \qquad \text{s.t. } \Phi X_0 = \Phi X. \tag{7}$$

## 3   Structured Dictionaries

It is now well-established that when the dictionary $\Phi$ is constructed with appropriate randomness, e.g., iid Gaussian entries, then for certain choices of $g$, in particular the convex selection $g(X) = \sum_i \|\boldsymbol{x}_{i\cdot}\|_2$ (which represents a generalization of the $\ell_1$ vector norm to row-sparse matrices), we can expect to recover $X_0$ exactly in the noiseless case or to close approximation otherwise. This assumes that $d(X_0)$ is sufficiently small relative to some function of the dictionary coherence or a related measure. However, with highly structured dictionaries these types of performance guarantees completely break down.

At the most basic level, one attempt to standardize structured dictionaries is by utilizing some form of column normalization as a pre-processing step. Most commonly, each column is scaled such that it has unit $\ell_2$ norm. This helps ensure that no one column is implicitly favored over another during the estimation process. However, suppose our observation matrix is generated via $Y = \Phi X_0$, where $\Phi = \widetilde{\Phi} D + \sigma \boldsymbol{a} \boldsymbol{b}^T$, $\widetilde{\Phi}$ is some well-behaved, incoherent dictionary, $D$ is a diagonal matrix, and $\sigma \boldsymbol{a} \boldsymbol{b}^T$ represents a rank one adjustment. If we apply column normalization to remove the effect of $D$, the resulting scale factors will be dominated by the rank one term when $\sigma$ is large. But if we do not column normalize, then $D$ can completely bias the estimation results.

In general, if our given dictionary is effectively $W\widetilde{\Phi} D$, with $W$ an arbitrary invertible matrix that scales and correlates rows, and $D$ diagonal, the combined effect can be severely disruptive. As an example from neuroimaging, the MEG/EEG source localization problem involves estimating sparse neural current sources within the brain using sensors placed near the surface of the scalp. The effective dictionary or forward model is characterized by highly correlated rows (because the sensors are physically constrained to be near one another) and columns with drastically different scales (since deep brain sources produce much weaker signals at the surface than superficial ones). More problematic is the situation where $\Phi = \widetilde{\Phi} S$, since an unrestricted matrix $S$ can introduce arbitrary coherence structure between individual or groups of columns in $\Phi$, meaning the structure of $\Phi$ is now arbitrary regardless of how well-behaved the original $\widetilde{\Phi}$.

## 4 Analysis

We will now analyze the properties of both Type I and Type II cost functions when coherent or highly structured dictionaries are present. Ideally, we would like to arrive at algorithms that are invariant, to the extent possible, to dictionary transformations that would otherwise disrupt the estimation efficacy. For simplicity, we will primarily consider the noiseless case, although we surmise that much of the underlying intuition carries over into the noiseless domain. This strategy mirrors the progression in the literature of previous sparse estimation theory related to the $\ell_1$ norm [3, 7, 8]. All proofs have been deferred to the Appendix, with some details omitted for brevity.

### 4.1 Invariance to $W$ and $D$

We will first consider the case where the observation matrix is produced via $Y = \Phi X_0 = W\widetilde{\Phi} D X_0$. Later in Sections 4.2 and 4.3 we will then address the more challenging situation where $\Phi = \widetilde{\Phi} S$.

**Lemma 1.** Let $W$ denote an arbitrary full-rank $n \times n$ matrix and $D$ an arbitrary full-rank $m \times m$ diagonal matrix. Then with $\alpha \to 0$, the Type II optimization problem

$$\min_X g^{(II)}(X) \qquad \text{s.t. } W\widetilde{\Phi} D X_0 = W\widetilde{\Phi} D X \tag{8}$$

is invariant to $W$ and $D$ in the sense that if $X^*$ is a global (or local) minimum to (8), then $D^{-1} X^*$ is a global (or local) minimum when we optimize $g^{(II)}(X)$ subject to the constraint $\widetilde{\Phi} X_0 = \widetilde{\Phi} X$.

Therefore, while switching between $\Phi = W\widetilde{\Phi} D$ and $\Phi = \widetilde{\Phi}$ may influence the initialization and possibly the update rules of a particular Type II algorithm, it does not fundamentally alter the underlying cost function. In contrast, Type I methods do not satisfy this invariance. Invariance is preserved with a $W$ factor in isolation. Likewise, inclusion of a $D$ factor alone with column normalization leads to invariance. However, inclusion of both $W$ and $D$ together can be highly disruptive.

Note that for improving Type I performance, it is not sufficient to apply some row decorrelating and normalizing $\hat{W}^{-1}$ to $\Phi$ and then column normalize with some $\hat{D}^{-1}$. This is because the application of $\hat{D}^{-1}$ will disrupt the effects of $\hat{W}^{-1}$. But one possibility to compensate for dictionary structure is to jointly learn a $\hat{W}^{-1}$ and $\hat{D}^{-1}$ that produces a $\Phi$ satisfying: (i) $\Phi\Phi^T = CI$ (meaning rows have a constant $\ell_2$ norm of $C$ and are uncorrelated, (ii) $\|\phi_{\cdot i}\|_2 = 1$ for all $i$. Up to irrelevant scale factors, a unique such transformation will always exist. In Section 5 we empirically demonstrate that this can be a highly effective strategy for improving the performance of Type I methods. However, as a final point, we should mention that the invariance Type II exhibits towards $W$ and $D$ (or any corrected form of Type I) will no longer strictly hold once noise is added.

## 4.2 Invariance to $S$: The $t > 1$ Case (Simultaneous Sparse Approximation)

We now turn to the potentially more problematic scenario with $\Phi = \widetilde{\Phi} S$. We will assume that $S$ is arbitrary with the only restriction being that the resulting $\Phi$ satisfies $\text{spark}[\Phi] = n + 1$, where matrix spark quantifies the smallest number of linearly dependent columns [7]. Consequently, the spark condition is equivalent to saying that each $n \times n$ sub-matrix of $\Phi$ is full rank. This relatively weak assumption is adopted for simplicity; in many cases it can be relaxed.

**Lemma 2.** Let $\Phi$ be an arbitrary dictionary with spark $[\Phi] = n + 1$ and $X_0$ a coefficient matrix with $d(X_0) < n$. Then there exists a constant $\rho > 0$ such that the optimization problem (7), with $g(X) = g^{(II)}(X)$ and $\alpha \to 0$, has no local minima and a unique, global solution at $X_0$ if $(\boldsymbol{x}_0)_{i\cdot}^T (\boldsymbol{x}_0)_{j\cdot} \leq \rho$ for all $i \neq j$ (i.e., the nonzero rows of $X_0$ are below some correlation threshold). Also, if we enforce exactly zero row-wise correlations, meaning $\rho = 0$, then a minimizing solution $X^*$ will satisfy $\|\boldsymbol{x}_{i\cdot}^*\|_2 = \|(\boldsymbol{x}_0)_{i\cdot}\|_2$ for all $i$ (i.e., a matching row-sparsity support), even for $d(X_0) \geq n$. This solution will be unique whenever $\Phi X_0 X_0^T \Phi = \Phi \Gamma \Phi^T$ has a unique solution for some non-negative, diagonal $\Gamma$.[2]

**Corollary 1.** There will always exist dictionaries $\Phi$ and coefficients $X_0$, consistent with the conditions from Lemma 2, such that the optimization problem (7) with any possible $g(X)$ of the form $g^{(I)}(X) = \sum_i h\left(\|\boldsymbol{x}_{i\cdot}\|_2\right)$ will have minimizing solutions not equal to $X_0$ (with or without column normalization).

In general, Lemma 2 suggests that for estimation purposes uncorrelated rows in $X_0$ can potentially compensate for troublesome dictionary structure, and together with Corollary 1 it also describes a potential advantage of Type II over Type I. Of course this result only stipulates sufficient conditions for recovery that are certainly not necessary, i.e., effective sparse recovery is possible even with correlated rows (more on this below). We also emphasize that the final property of Lemma 2 implies that the row norms of $X_0$ (and therefore the row-sparsity support) can still be recovered even up to the extreme case of $d(X_0) = m > n$. While this may seem surprising at first, especially since even brute force minimization of (3) can not achieve a similar feat, it is important to keep in mind that (3) is blind to the correlation structure of $X_0$. Although Type II does not explicitly require any such structure, it is able to outperform (3) by implicitly leveraging this structure when the situation happens to be favorable. While space prevents a full treatment, in the context of MEG/EEG source estimation, we have successfully localized 500 nonzero sources (rows) using a $100 \times 1000$ dictionary.

However, what about the situation where strong correlations do exist between the nonzero rows of $X_0$? A couple things are worth mentioning in this regard. First, Lemma 2 can be strengthened considerably via the expanded optimization problem: $\min_{X,B} g^{(II)}(X)$ s.t. $\Phi X_0 = \Phi X B$, which achieves a result similar to Lemma 2 but with a weaker correlation condition (although the row-norm recovery property is lost). Secondly, in the case of perfect correlation between rows (the hardest case), the problem reduces to an equivalent one with $t = 1$, i.e., it exactly reduces to the canonical sparse recovery problem. We address this situation next.

## 4.3 Invariance to $S$: The $t = 1$ Case (Standard Sparse Approximation)

This section considers the $t = 1$ case, meaning $Y = \boldsymbol{y}$ and $X_0 = \boldsymbol{x}_0$ are now vectors. For convenience, we define $\mathcal{X}(\mathcal{S}, \mathcal{P})$ as the set of all coefficient vectors in $\mathbb{R}^m$ with support (or nonzero coefficient locations) specified by the index set $\mathcal{S} \subset \{1, \ldots, m\}$ and sign pattern given by $\mathcal{P} \in \{-1, +1\}^{|\mathcal{S}|}$ (here the $|\cdot|$ operator denotes the cardinality of a set).

**Lemma 3.** Let $\Phi$ be an arbitrary dictionary with spark $[\Phi] = n + 1$. Then for any $\mathcal{X}(\mathcal{S}, \mathcal{P})$ with $|\mathcal{S}| < n$, there exists a non-empty subset $\bar{\mathcal{X}} \subset \mathcal{X}(\mathcal{S}, \mathcal{P})$ (with nonzero Lebesgue measure), such that if $\boldsymbol{x}_0 \in \bar{\mathcal{X}}$, the Type II minimization problem

$$\min_{\boldsymbol{x}} g^{(II)}(\boldsymbol{x}) \qquad \text{s.t. } \Phi \boldsymbol{x}_0 = \Phi \boldsymbol{x}, \alpha \to 0 \tag{9}$$

will have a unique minimum and it will be located at $\boldsymbol{x}_0$.

This Lemma can be obtained with a slight modification of results in [18]. In other words, no matter how poorly structured a particular dictionary is with regard to a given sparsity profile, there will always be sparse coefficients we are guaranteed to recover (provided we utilize a convergent algorithm). In contrast, an equivalent claim can not be made for Type I:

**Lemma 4.** Given an arbitrary Type I penalty $g^{(I)}(\boldsymbol{x}) = \sum_i h(|x_i|)$, with $h$ a fixed, non-decreasing function, there will always exist a dictionary $\Phi$ (with or without normalized columns) and set $\mathcal{X}(\mathcal{S}, \mathcal{P})$ such that for any $\boldsymbol{x}_0 \in \mathcal{X}(\mathcal{S}, \mathcal{P})$, the problem

$$\min_{\boldsymbol{x}} g^{(I)}(\boldsymbol{x}) \qquad \text{s.t. } \Phi\boldsymbol{x}_0 = \Phi\boldsymbol{x} \tag{10}$$

will not have a unique minimum located at $\boldsymbol{x}_0$.

This can happen because the global minimum does not equal $\boldsymbol{x}_0$ and/or because of the presence of local minima. Of course this does not necessarily imply that a particular Type I algorithm will fail. For example, even with multiple minima, an appropriate optimization strategy could conceivably still locate an optimum that coincides with $\boldsymbol{x}_0$. While it is difficult to analyze all possible algorithms, we can address one influential variety based on iterative reweighted $\ell_1$ minimization [4, 18]. Here the idea is that if $h$ is concave and differentiable, then a convergent means of minimizing (10) is to utilize a first-order Taylor series approximation of $g^{(I)}(\boldsymbol{x})$ at some point $\hat{\boldsymbol{x}}$. This leads to an iterative procedure where at each step we must first compute $h_i' \triangleq dh(z)/dz|_{z=|\hat{x}_i|}$ and then minimize $\sum_i h_i'|x_i|$ subject to $\Phi\boldsymbol{x}_0 = \Phi\boldsymbol{x}$ to update $\hat{\boldsymbol{x}}$. This method produces a sparse estimate at each iteration and is guaranteed to converge to a local minima (or stationary point) of (10). However, this solution may be suboptimal in the following sense:

**Corollary 2.** Given an arbitrary $g^{(I)}(\boldsymbol{x})$ as in Lemma 4, there will always exist a $\Phi$ and $\mathcal{X}(\mathcal{S}, \mathcal{P})$, such that for any $\boldsymbol{x}_0 \in \mathcal{X}(\mathcal{S}, \mathcal{P})$, iterative reweighted $\ell_1$ minimization will not converge to $\boldsymbol{x}_0$ when initialized at the minimum $\ell_1$ norm solution.

Note that this failure does not result from a convergence pathology. Rather, the presence of minima different from $\boldsymbol{x}_0$ explicitly disrupts the algorithm.

In general, with highly structured dictionaries deviating from the ideal, the global minimum of convex penalties often does not correspond with $\boldsymbol{x}_0$ as theoretical equivalence results break down. This in turn suggests the use of concave penalty functions to seek possible improvement. However, as illustrated by the following result, even the simplest of sparse recovery problems, that of estimating some $\boldsymbol{x}_0$ with only one nonzero element using a dictionary with a 1D null-space, Type I can be characterized by problematic local minima with (strictly) concave penalties. For this purpose we define $\boldsymbol{\phi}_*$ as an arbitrary column of $\Phi$ and $\bar{\Phi}_*$ as all columns of $\Phi$ excluding $\boldsymbol{\phi}_*$.

**Lemma 5.** Let $h$ denote a concave, non-decreasing function with $h_{max}' \triangleq \lim_{z\to 0} dh(z)/dz$ and $h_{min}' \triangleq \lim_{z\to\infty} dh(z)/dz$. Also, let $\Phi$ be a dictionary with unit $\ell_2$ norm columns and spark $[\Phi] = m = n + 1$ (i.e., a 1D null-space), and let $\boldsymbol{x}_0$ satisfy $\|\boldsymbol{x}_0\|_0 = 1$ with associated $\boldsymbol{\phi}_*$. Then the Type I problem (10) can have multiple local minima if

$$\frac{h_{max}'}{h_{min}'} > \|\bar{\Phi}_*^{-1}\boldsymbol{\phi}_*\|_1. \tag{11}$$

This result has a very clear interpretation related to how dictionary coherence can potentially disrupt even the most rudimentary of estimation tasks. The righthand side of (11) is bounded from below by 1, which is approached whenever one or more columns in some $\bar{\Phi}_*$ are similar to $\boldsymbol{\phi}_*$ (i.e., coherent). Thus, even the slightest amount of curvature (or strict concavity) in $h$ can lead to the inequality being satisfied when highly coherent columns are present. While obviously with $h(z) = z$ this will not be an issue (consistent with the well-known convexity of the $\ell_1$ problem), for many popular non-convex penalties, this gradient ratio may be large relative to the righthand side, indicating that local minima

are always possible. For example, with the $h(z) = \log(z + \alpha)$ selection from [4] $h'_{min} \to 0$ for all $\alpha$ while $h'_{max} \to 1/\alpha$. We note that Type II has provably no local minima in this regime (this follows as a special case of Lemma 3). Of course the point here is not that Type I algorithms are incapable of solving simple problems with $\|\boldsymbol{x}_0\|_0 = 1$ (and any iterative reweighted $\ell_1$ scheme will succeed on the first step anyway). Rather, Lemma 5 merely demonstrates how highly structured dictionaries can begin to have negative effects on Type I, potentially more so than with Type II, even on trivial tasks. The next section will empirically explore this conjecture.

## 5   Empirical Results

We now present two simulation examples illustrating the potential benefits of Type II with highly structured dictionaries. In the first experiment, the dictionary represents an MEG leadfield, which at a high level can be viewed as a mapping from the electromagnetic (EM) activity within $m$ brain voxels to $n$ sensors placed near the scalp surface. Computed using Maxwell's equations and a spherical shell head model [12], the resulting $\Phi$ is characterized by highly correlated rows, because the small scalp surface requires that sensors be placed close together, and vastly different column norms, since the EM field strength drops off rapidly for deep brain sources. These effects are well represented by a dictionary such as $\Phi = W\widetilde{\Phi}D$ as discussed previously. Figure 1 (*Left*) displays trial-averaged results comparing Type I algorithms with Type II using such an MEG leadfield dictionary. Data generation proceeded as follows: We produce $\Phi$ by choosing 50 random sensor locations and 100 random voxels within the brain volume. We then create a coefficient matrix $X_0$ with $t = 5$ columns and $d(X_0)$ an experiment-dependent parameter. Nonzero rows of $X_0$ are drawn iid from a unit Gaussian distribution. The observation matrix is then computed as $Y = \Phi X_0$. We run each algorithm and attempt to estimate $X_0$, calculating the probability of success averaged over 200 trials as $d(X_0)$ is varied from 10 to 50. We compared Type II, implemented via a simple iterative reweighted $\ell_2$ approach, with two different Type I schemes. The first is a homotopy continuation method using the Type I penalty $g^{(I)}(X) = \sum_i \log(\|\boldsymbol{x}_{i\cdot}\|_2^2 + \alpha)$, where $\alpha$ is gradually reduced to zero during the estimation process [5]. We have often found this to be the near optimal Type I approach on a variety empirical tests. Secondly, we used the standard mixed-norm penalty $g^{(I)}(X) = \|X\|_{1,2} = \sum_i \|\boldsymbol{x}_{i\cdot}\|_2$, which leads to a convex minimization problem that generalizes basis pursuit (or the lasso), to the $t > 1$ domain [6, 10].

While Type II displays invariance to $W$- and $D$-like transformations, Type I methods do not. Consequently, we examined two dictionary-standardization methods for Type I. First, we utilized basic $\ell_2$ column normalization, without which Type I will have difficulty with the vastly different column scalings of $\Phi$. Secondly, we developed an algorithm to learn a transformed dictionary $\hat{U}\Phi\hat{\Pi}$, with $\hat{U}$ arbitrary, $\hat{\Pi}$ diagonal, such that the combined dictionary has uncorrelated, unit $\ell_2$ norm rows, and unit $\ell_2$ norm columns (as discussed in Section 4.1). Figure 1(*left*) contains results from all of these variants, where it is clear that some compensation for the dictionary structure is essential for good recovery performance. We also note that Type II still outperforms Type I in all cases, suggesting that even after transformation of the latter, there is still residual structure in the MEG leadfield being exploited by Type II. This is a very reasonable assumption given that $\Phi$ will typically have strong column-wise correlations as well, which are more effectively modeled by right multiplication by some $S$. As a final point, the Type II success probability does not go to zero even when $d(X_0) = 50$, implying that in some cases it is able to find a number of nonzeros equal to the number of rows in $\Phi$. This is possible because even with only $t = 5$ columns, the nonzero rows of $X_0$ display somewhat limited sample correlation, and so exact support recovery is still possible. With $t > 5$ these sample correlations can be reduced further, allowing consistent support recovery when $d(X_0) > n$ (not shown).

To further test the ability of Type II to handle structure imposed by some $\widetilde{\Phi}S$, we performed a second experiment with explicitly controlled correlations among groups of columns. For each trial we generated a $50 \times 100$ Gaussian iid dictionary $\widetilde{\Phi}$. Correlations were then introduced using a block-diagonal $S$ with $4 \times 4$ blocks created with iid entries drawn from a uniform distribution (between 0 and 1). The resulting $\Phi = \widetilde{\Phi}S$ was then scaled to have unit $\ell_2$ norm columns. We then generated a random $\boldsymbol{x}_0$ vector ($t = 1$ case) using iid Gaussian nonzero entries with $d(\boldsymbol{x}_0)$ varied from 10 to 25 (with $t = 1$, we cannot expect to recover as many nonzeros as when $t = 5$). Signal vectors are computed as $\boldsymbol{y} = \Phi\boldsymbol{x}_0$ or, for purposes of direct comparison with a canonical iid dictionary, $\boldsymbol{y} = \widetilde{\Phi}\boldsymbol{x}_0$. We evaluated Type II and the Type I iterative reweighted $\ell_1$ minimization

method from [4], which is guaranteed to do as well or better than standard $\ell_1$ norm minimization. Trial-averaged results using both $\Phi$ and $\widetilde{\Phi}$ are shown in Figure 1(*right*), where it is clear that while Type II performance is essentially unchanged, Type I performance degrades substantially.

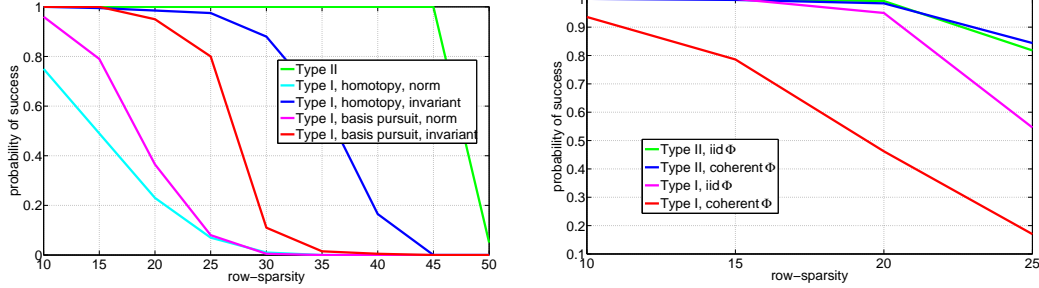

Figure 1: *Left*: Probability of success recovering coefficient matrices with varying degrees of row-sparsity using an MEG leadfield as the dictionary. Two Type I methods were compared, a homotopy continuation method from [5] and a version of basis pursuit extended to the simultaneous sparse approximation problem by minmizing the $\ell_{1,2}$ mixed norm [6, 10]. Type I methods were compared using standard $\ell_2$ column normalization and a learned invariance transformation. *Right*: Probability of success recovering sparse vectors using a Gaussian iid dictionary $\widetilde{\Phi}$ and a coherent dictionary $\Phi$ with clustered columns. The Type I method was the interactive reweighted $\ell_1$ algorithm from [4].

## 6   Conclusion

When we are free to choose the basis vectors of an overcomplete signal dictionary, the sparse estimation problem is supported by strong analytical and empirical foundations. However, there are many applications where physical restrictions or other factors impose rigid constraints on the dictionary structure such that the assumptions of theoretical recovery guarantees are violated. Examples include model selection problems with correlated features, source localization, and compressive sensing with constrained measurement directions. This can have significant consequences depending on how the estimated coefficients will ultimately be utilized. For example, in the source localization problem, correlated dictionary columns may correspond with drastically different regions (e.g., brain areas), so recovering the exact sparsity profile can be important. Ideally we would like our recovery algorithms to display invariance, to the extent possible, to the actual structure of the dictionary. With typical Type I sparsity penalties this can be a difficult undertaking; however, with the natural dictionary dependence of the Type II penalty, to some extent it appears this structure can be accounted for, leading to more consistent performance across dictionary types.

## Appendix

Here we provide brief proofs of several results from the paper. Some details have been omitted for space considerations.

***Proof of Lemma 1***: First we address invariance with respect to $W$. Obviously the equality constraint is unaltered by a full rank $W$, so it only remains to check that the dictionary-dependent penalty $g^{(II)}$ is invariant. However, since by standard determinant relationships $\log |W\widetilde{\Phi}D\Gamma D\widetilde{\Phi}^T W^T| = \log |W||\widetilde{\Phi}D\Gamma D\widetilde{\Phi}^T||W^T| = \log |\widetilde{\Phi}D\Gamma D\widetilde{\Phi}^T| + C$, where $C$ is an irrelevant constant for optimization purposes, this point is established. With respect to $D$, we re-parameterize the problem by defining $\widetilde{X} \triangleq DX$ and $\widetilde{\Gamma} \triangleq D\Gamma D$. It is then readily apparent that the penalty (6) satisfies

$$g^{(II)}(X) \equiv \min_{\Gamma \succeq 0} \text{Tr}\left[X^T \Gamma^{-1} X\right] + \log |\widetilde{\Phi}D\Gamma D\widetilde{\Phi}^T| = \min_{\widetilde{\Gamma} \succeq 0} \text{Tr}\left[\widetilde{X}^T \widetilde{\Gamma}^{-1} \widetilde{X}\right] + \log |\widetilde{\Phi}\widetilde{\Gamma}\widetilde{\Phi}^T|. \quad (12)$$

So we are effectively solving: $\min_{\widetilde{X}} g^{(II)}(\widetilde{X})$ s.t. $\widetilde{\Phi}DX_0 = \widetilde{\Phi}\widetilde{X}$. ∎

***Proof of Lemma 2 and Corollary 1***: Minimizing the Type II cost function can be accomplished equivalently by minimizing

$$\mathcal{L}(\Gamma) \triangleq \mathrm{Tr}\left[\Phi t^{-1} X_0 X_0^T \Phi^T \left(\Phi\Gamma\Phi^T\right)^{-1}\right] + \log|\Phi\Gamma\Phi^T|, \tag{13}$$

over the non-negative diagonal matrix $\Gamma$ (this follows from a duality principle in Type II models [18]). $\mathcal{L}(\Gamma)$ includes an observed covariance $\Phi t^{-1} X_0 X_0^T \Phi^T$ and a parameterized model covariance $\Phi\Gamma\Phi^T$, and is globally minimized with $\Gamma^* = t^{-1}\mathrm{diag}[X_0 X_0^T]$ [17]. Moreover, if $\Phi\Gamma^*\Phi^T$ is sufficiently close to $t^{-1}\Phi X_0 X_0^T \Phi^T$, meaning the off-diagonal elements of $X_0 X_0^T$ are not too large, then it can be shown by differentiating along the direction between any arbitrary point $\Gamma'$ and $\Gamma^*$ that no local minima exist, leading to the first part of Lemma 2.

Regarding the second part, we now allow $d(X_0)$ to be arbitrary but require that $X_0 X_0^T$ be diagonal (zero correlations). Using similar arguments as above, it is easily shown that any minimizing solution $\Gamma^*$ must satisfy $\Phi\Gamma^*\Phi^T = \Phi t^{-1} X_0 X_0^T \Phi^T$. This equality can be viewed as $n(n+1)/2$ linear equations (equal to the number of unique elements in an $n \times n$ covariance matrix) and $m$ unknowns, namely, the diagonal elements of $\Gamma^*$. Therefore, if $n(n+1)/2 > m$ this system of equations will typically be overdetermined (e.g., if suitable randomness is present to avoid adversarial conditions) with a unique solution. Moreover, because of the requirement that $\Gamma$ be non-negative, it is likely that a unique solution will exist in many cases where $m$ is even greater than $n(n+1)/2$ [2].

Finally, we address Corollary 1. First, consider the case where $t = 1$, so $X_0 = \boldsymbol{x}_0$. To satisfy the now degenerate correlation condition, we must have $d(\boldsymbol{x}_0) = 1$. Even in this simple regime it can be demonstrated that a unique minimum at $\boldsymbol{x}_0$ is possible iff $h(z) = z$ based on Lemma 5 (below) and a complementary result in [17]. So the only Type I possibility is $h(z) = z$. A simple counterexample with $t = 2$ serves to rule this selection out. Consider a dictionary $\Phi$ and two coefficient matrices given by

$$\Phi = \begin{bmatrix} \epsilon & \epsilon & 1 & 1 \\ 1 & -1 & 0 & 0 \\ 0 & 0 & \epsilon & -\epsilon \end{bmatrix}, \quad X_{(1)} = \begin{bmatrix} 1 & 1 \\ 1 & -1 \\ 0 & 0 \\ 0 & 0 \end{bmatrix}, \quad X_{(2)} = \begin{bmatrix} 0 & 1 \\ 0 & -1 \\ \epsilon & 0 \\ \epsilon & 0 \end{bmatrix}, \tag{14}$$

It is easily verified that $\Phi X_{(1)} = \Phi X_{(2)}$ and that $X_{(1)} = X_0$, the maximally row-sparse solution. Computing the Type I cost for each with $h(z) = z$ gives $g^{(I)}(X_{(1)}) = 2\sqrt{2}$ and $g^{(I)}(X_{(2)}) = 2(1 + \epsilon)$. Thus, if we allow $\epsilon$ to be small, $g^{(I)}(X_{(2)}) < g^{(I)}(X_{(1)})$, so $X_{(1)} = X_0$ cannot be the minimizing solution. Note that $\ell_2$ column normalization will not change this conclusion since all columns of $\Phi$ have equal norm already. ∎

***Proof of Lemma 4 and Corollary 2***: For brevity, we will assume that $h$ is concave and differentiable, as is typical of most sparsity penalties used in practice (the more general case follows with some additional effort). This of course includes $h(z) = z$, which is both concave and convex, and leads to the $\ell_1$ norm penalty. These results will now be demonstrated using a simple counterexample similar to the one above. Assume we have the dictionary $\Phi$ from (14), and that $\mathcal{S} = \{1, 2\}$ and $\mathcal{P} = \{+1, +1\}$, which implies that any $\boldsymbol{x}_0 \in \mathcal{X}(\mathcal{S}, \mathcal{P})$ can be expressed as $\boldsymbol{x}_0 = [\alpha_1, \alpha_2, 0, 0]^T$, for some $\alpha_1, \alpha_2 > 0$. We will now show that with any member from this set, there will not be a unique minimum to the Type I cost at $\boldsymbol{x}_0$ for any possible concave, differentiable $h$.

First assume $\alpha_1 \geq \alpha_2$. Consider the alternative feasible solution $\boldsymbol{x}_{(2)} = [(\alpha_1 - \alpha_2), 0, \epsilon\alpha_2, \epsilon\alpha_2]^T$. To check if this is a local minimum, we can evaluate the gradient of the penalty function $g^{(I)}(\boldsymbol{x})$ along the feasible region near $\boldsymbol{x}_{(2)}$. Given $\boldsymbol{v} = [1, 1, -\epsilon, -\epsilon]^T \in \mathrm{Null}(\Phi)$, this can be accomplished by computing $\partial g^{(I)}(\boldsymbol{x}_{(2)} + \beta\boldsymbol{v})/\partial\beta = h'(|\alpha_1 - \alpha_2 + \beta|) + h'(|\beta|) + 2\epsilon h'(|\epsilon\alpha_2 - \epsilon\beta|)$. In the limit as $\beta \to 0$ (from the right or left), this expression will always be positive for $\epsilon < 0.5$ based on the concavity of $h$. Therefore, $\boldsymbol{x}_{(2)}$ must be a minimum. By symmetry an equivalent argument can be made when $\alpha_2 \geq \alpha_1$. (In the special case where $\alpha_1 = \alpha_2$, there will actually exist two maximally sparse solutions, the generating $\boldsymbol{x}_0$ and $\boldsymbol{x}_{(2)}$.) It is also straightforward to verify analytically that iterative reweighted $\ell_1$ minimization will fail on this example when initialized at the minimum $\ell_1$ norm solution. It will always become trapped at $\boldsymbol{x}_{(2)}$ after the first iteration, assuming $\alpha_1 \geq \alpha_2$, or a symmetric local minimum otherwise. ∎

***Proof of Lemma 5***: This result can be shown by examining properties of various gradients along the feasible region, not unlike some of the analysis above, and then bounding the resultant quantity. We defer these details to a later publication. ∎

## Footnotes

*Draft version for NIPS 2011 pre-proceedings.

[1]Other row norms, such as the $\ell_\infty$, have been considered as well but are less prevalent.

[2]See Appendix for more details about this condition. In most situations, it will hold if $m < n(n+1)/2$, and likely for many instances with $m$ even greater than this.

# References

[1] S. Baillet, J.C. Mosher, and R.M. Leahy, "Electromagnetic brain mapping," *IEEE Signal Processing Magazine*, pp. 14–30, Nov. 2001.

[2] A.M. Bruckstein, M. Elad, and M. Zibulevsky, "A non-negative and sparse enough solution of an underdetermined linear system of equations is unique," *IEEE Trans. Information Theory*, vol. 54, no. 11, pp. 4813–4820, Nov. 2008.

[3] E. Candès, J. Romberg, and T. Tao, "Robust uncertainty principles: Exact signal reconstruction from highly incomplete frequency information," *IEEE Trans. Information Theory*, vol. 52, no. 2, pp. 489–509, Feb. 2006.

[4] E. Candès, M. Wakin, and S. Boyd, "Enhancing sparsity by reweighted $\ell_1$ minimization," *J. Fourier Anal. Appl.*, vol. 14, no. 5, pp. 877–905, 2008.

[5] R. Chartrand and W. Yin, "Iteratively reweighted algorithms for compressive sensing," *Proc. Int. Conf. Accoustics, Speech, and Signal Proc.*, 2008.

[6] S.F. Cotter, B.D. Rao, K. Engan, and K. Kreutz-Delgado, "Sparse solutions to linear inverse problems with multiple measurement vectors," *IEEE Trans. Signal Processing*, vol. 53, no. 7, pp. 2477–2488, April 2005.

[7] D.L. Donoho and M. Elad, "Optimally sparse representation in general (nonorthogonal) dictionaries via $\ell_1$ minimization," *Proc. National Academy of Sciences*, vol. 100, no. 5, pp. 2197–2202, March 2003.

[8] J.J. Fuchs, "On sparse representations in arbitrary redundant bases," *IEEE Trans. Information Theory*, vol. 50, no. 6, pp. 1341–1344, June 2004.

[9] S. Ji, D. Dunson, and L. Carin, "Multi-task compressive sensing," *IEEE Trans. Signal Processing*, vol. 57, no. 1, pp. 92–106, Jan 2009.

[10] D.M. Malioutov, M. Çetin, and A.S. Willsky, "Sparse signal reconstruction perspective for source localization with sensor arrays," *IEEE Transactions on Signal Processing*, vol. 53, no. 8, pp. 3010–3022, August 2005.

[11] B.D. Rao, K. Engan, S. F. Cotter, J. Palmer, and K. Kreutz-Delgado, "Subset selection in noise based on diversity measure minimization," *IEEE Trans. Signal Processing*, vol. 51, no. 3, pp. 760–770, March 2003.

[12] J. Sarvas, "Basic methematical and electromagnetic concepts of the biomagnetic inverse problem," *Phys. Med. Biol.*, vol. 32, pp. 11–22, 1987.

[13] J.G. Silva, J.S. Marques, and J.M. Lemos, "Selecting landmark points for sparse manifold learning," *Advances in Neural Information Processing Systems 18*, pp. 1241–1248, 2006.

[14] M.E. Tipping, "Sparse Bayesian learning and the relevance vector machine," *Journal of Machine Learning Research*, vol. 1, pp. 211–244, 2001.

[15] J.A. Tropp, "Algorithms for simultaneous sparse approximation. Part II: Convex relaxation," *Signal Processing*, vol. 86, pp. 589–602, April 2006.

[16] M.B. Wakin, M.F. Duarte, S. Sarvotham, D. Baron, and R.G. Baraniuk, "Recovery of jointly sparse signals from a few random projections," *Advances in Neural Information Processing Systems 18*, pp. 1433–1440, 2006.

[17] D.P. Wipf, *Bayesian Methods for Finding Sparse Representations*, PhD Thesis, University of California, San Diego, 2006.

[18] D.P. Wipf and S. Nagarajan, "Iterative reweighted $\ell_1$ and $\ell_2$ methods for finding sparse solutions," *J. Selected Topics in Signal Processing (Special Issue on Compressive Sensing)*, vol. 4, no. 2, April 2010.

